# Visual Recognition using Embedded Feature Selection for Curvature Self-Similarity

**Angela Eigenstetter**
HCI & IWR, University of Heidelberg
aeigenst@iwr.uni-heidelberg.de

**Björn Ommer**
HCI & IWR, University of Heidelberg
ommer@uni-heidelberg.de

## Abstract

Category-level object detection has a crucial need for informative object representations. This demand has led to feature descriptors of ever increasing dimensionality like co-occurrence statistics and self-similarity. In this paper we propose a new object representation based on curvature self-similarity that goes beyond the currently popular approximation of objects using straight lines. However, like all descriptors using second order statistics, ours also exhibits a high dimensionality. Although improving discriminability, the high dimensionality becomes a critical issue due to lack of generalization ability and curse of dimensionality. Given only a limited amount of training data, even sophisticated learning algorithms such as the popular kernel methods are not able to suppress noisy or superfluous dimensions of such high-dimensional data. Consequently, there is a natural need for feature selection when using present-day informative features and, particularly, curvature self-similarity. We therefore suggest an embedded feature selection method for SVMs that reduces complexity and improves generalization capability of object models. By successfully integrating the proposed curvature self-similarity representation together with the embedded feature selection in a widely used state-of-the-art object detection framework we show the general pertinence of the approach.

## 1 Introduction

One of the key challenges of computer vision is the robust representation of complex objects and so over the years, increasingly rich features have been proposed. Starting with brightness values of image pixels and simple edge histograms [10] descriptors evolved and more sophisticated features like shape context [1] and wavelets [23] were suggested. The probably most widely used and best performing image descriptors today are SIFT [18] and HOG [4] which model objects based on edge orientation histograms. Recently, there has been a trend to utilize more complicated image statistics like co-occurrence and self-similarity [25, 5, 15, 29, 31] to build more robust descriptors. This development shows, that the dimensionality of descriptors is getting larger and larger. Furthermore it is noticeable that all descriptors that model the object boundary rely on image statistics that are primarily based on edge orientation. Thus, they approximate objects with straight lines. However, it was shown in different studies within the perception community that besides orientation also curvature is an important cue when performing visual search tasks. In our earlier work [21] we extended the modeling of object boundary contours beyond the widely used edge orientation histograms by utilizing curvature information to overcome the drawbacks of straight line approximations. However, curvature can provide even more information about the object boundary. By computing co-occurrences between discriminatively curved boundaries we build a curvature self-similarity descriptor that provides a more detailed and accurate object description. While it was shown that self-similarity and co-occurrence lead to very robust and highly discriminative object representations, these second order image statistics are also pushing feature spaces to extremely

high dimensions. Since the amount of training data stays more or less the same, the dimensionality of the object representation has to be reduced to prevent systems to suffer from curse of dimensionality and overfitting. Nevertheless, well designed features still increase performance. Deselaers et al. [5], for instance, suggested an approach that results in a 160000 dimensional descriptor which was evaluated on the ETHZ shape dataset which contains on average 30 positive object instances per category. To exploit the full capabilities of high-dimensional representations applied in object detection we developed a new embedded feature selection method for SVM which reliable discards superfluous dimensions and therefore improves object detection performance.

The paper is organized as follows: First we will give a short overview on embedded feature selection methods for SVMs (Section 2.1) and describe a novel method to capture the important dimensions from high-dimensional representations (Section 2.2). After that we describe our new self-similarity descriptor based on curvature to go beyond the straight line approximation of objects to a more accurate description (Section 3). Moreover, Section 3 discusses previous work on self-similarity. In the experimental section at the end of the paper we evaluate the suggested curvature self-similarity descriptor along with our feature selection method.

## 2 Feature Selection for Support Vector Machines

### 2.1 Embedded Feature Selection Approaches

Guyon et al. [12] categorize feature selection methods into filters, wrappers and embedded methods. Contrary to filters and wrappers embedded feature selection methods incorporate feature selection as a part of the learning process (for a review see [17]). The focus of this paper is on embedded feature selection methods for SVMs, since most state-of-the-art detection systems use SVM as a classifier. To directly integrate feature selection into the learning process of SVMs sparsity can be enforced on the model parameter $\mathbf{w}$. Several researchers e.g [2] have considered replacing the L2 regularization term $\|\mathbf{w}\|_2^2$ with an L1 regularization term $\|\mathbf{w}\|_1$. Since L1 norm penalty for SVM has some serious limitations, Wang et al. [30] suggested the doubly regularized SVM (DrSVM) which is not replacing the L2 regularization but adding an additional L1 regularization to automatically select dimensions during the learning process.

Contrary to linear SVM enforcing sparsity on the model parameter $\mathbf{w}$ does reduce dimensionality for non-linear kernel functions in the higher dimensional kernel space rather than in the number of input features. To reduce the dimensionality for non-linear SVMs in the feature space one can introduce an additional selection vector $\boldsymbol{\theta} \in [0,1]^n$, where larger values of $\theta_i$ indicate more useful features. The objective is then to find the best kernel of the form $K_{\boldsymbol{\theta}}(\mathbf{x}, \mathbf{z}) = K(\boldsymbol{\theta} * \mathbf{x}, \boldsymbol{\theta} * \mathbf{z})$, where $\mathbf{x}, \mathbf{z} \in \mathbb{R}^n$ are the feature vectors and $*$ is element-wise multiplication. These hyper-parameters $\boldsymbol{\theta}$ can be obtained via gradient descent on a generalization bound or a validation error. Another possibility is to consider the scaling factors $\boldsymbol{\theta}$ as parameters of the learning algorithm [11], where the problem was solved using a reduced conjugate gradient technique.

In this paper we integrate the scaling factors into the learning algorithm, but instead of using L2 norm constraint like in [11] on the scaling parameter $\boldsymbol{\theta}$ we apply an L1 norm sparsity which is explicitly discarding dimensions of the input feature vector. For the linear case our optimization problem becomes similar to DrSVM [30] where a gradient descent method is applied to find the optimal solution $\mathbf{w}^*$. To find a starting point a computational costly initialization is applied, while our selection step can start at the canonical $\boldsymbol{\theta} = \mathbf{1}$, because $\mathbf{w}$ is modeled in a separate variable.

### 2.2 Iterative Dimensionality Reduction for SVM

A SVM classifier is learning a hyperplane defined by $\mathbf{w}$ and $b$ which best separates the training data $\{(\mathbf{x}_i, y_i)\}_{1 \leq i \leq N}$ with labels $y_i \in \{-1, +1\}$. We are following the concept of embedded feature selection and therefore include the feature selection parameter $\boldsymbol{\theta}$ directly in the SVM classifier. The corresponding optimization problem can be expressed in the following way:

$$\min_{\boldsymbol{\theta}} \min_{\mathbf{w}, b, \boldsymbol{\xi}} \quad \frac{1}{2}\|\mathbf{w}\|_2^2 + C \sum_{i=1}^{N} \xi_i \tag{1}$$

$$\text{subject to}: \quad y_i(\mathbf{w}^T \psi(\boldsymbol{\theta} * \mathbf{x}_i) + b) \geq 1 - \xi_i \quad \wedge \quad \xi_i \geq 0 \quad \wedge \quad \|\boldsymbol{\theta}\|_1 \leq \theta_0$$

Algorithm 1: Iterative Dimensionality Reduction for SVM

1: converged := FALSE, $\boldsymbol{\theta}$ := 1
2: **while** converged==FALSE **do**
3:     [$\mathbf{x}'_l$, $\alpha$, b] = trainSVM( $X'$, $Y'$, $\boldsymbol{\theta}$, C)
4:     $\boldsymbol{\theta}^*$ = applyBundleMethod($X''$,$Y''$,$\mathbf{x}'_l$,$\alpha$,b,C)
5:     **if** $\boldsymbol{\theta}^*$ == $\boldsymbol{\theta}$ **then**
6:         converged=TRUE;
7:     **end if**
8:     $\boldsymbol{\theta}$ = $\boldsymbol{\theta}^*$
9: **end while**

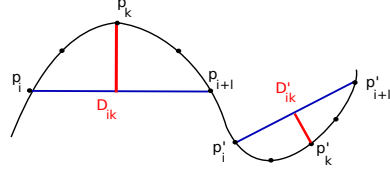

Figure 1: Visualization of curvature computation. $D_{ik}$ is on the left-hand side of the vector $(p_{i+l} - p_i)$ and therefore has a positive sign, while $D'_{ik}$ is on the right-hand side of the vector $(p'_{i+l} - p'_i)$ and therefore gets a negative sign

where $K(\mathbf{x}, \mathbf{z}) := \psi(\mathbf{x}) \cdot \psi(\mathbf{z})$ is the SVM kernel function. The function $\psi(\mathbf{x})$ is typically unknown and represents the mapping of the feature vector $\mathbf{x}$ into a higher dimensional space. We enforce sparsity of the feature selection parameter $\boldsymbol{\theta}$ by the last constraint of Eq. 1, which restricts the L1-norm of $\boldsymbol{\theta}$ by a constant $\theta_0$. Since SVM uses L2 normalization it does not explicitly enforce single dimensions to be exactly zero. However, this is necessary to explicitly discard unnecessary dimensions. We rewrite the problem in Eq. 1 without additional constraints in the following way:

$$\min_{\boldsymbol{\theta}} \min_{\mathbf{w},b} \lambda\|\boldsymbol{\theta}\|_1 + \frac{1}{2}\|\mathbf{w}\|_2^2 + C\sum_{i=1}^{N}\max(0, 1 - y_i f_{\boldsymbol{\theta}}(\mathbf{x}_i)) \tag{2}$$

where the decision function $f_{\boldsymbol{\theta}}$ is given by $f_{\boldsymbol{\theta}}(\mathbf{x}) = \mathbf{w}^T\psi(\boldsymbol{\theta} * \mathbf{x}) + b$. Note, that the last constraint, where the L1-norm is restricted by a constant $\theta_0$ is rewritten as an L1-regularization term, multiplied with the sparsity parameter $\lambda$.

Due to the complexity of problem 2 we propose to solve two simpler problems iteratively. We first split the training data into three sets, training $\{(\mathbf{x}'_i, y'_i)\}_{1 \leq i \leq N'}$, validation $\{(\mathbf{x}''_i, y''_i)\}_{1 \leq i \leq N''}$ and a hold out testset. Now we optimize the problem according to $\mathbf{w}$ and $b$ for a fixed selection parameter $\boldsymbol{\theta}$ using a standard SVM algorithm on the training set. Parameter $\boldsymbol{\theta}$ is optimized in a second optimization step on the validation data using an extended version of the bundle method suggested in [6]. We are performing the second step of our algorithm on a separate validation set to prevent overfitting. In the first step of our algorithm, the parameter $\boldsymbol{\theta}$ is fixed and the remaining problem is converted into the dual problem

$$\max_{\boldsymbol{\alpha}} \sum_{i=1}^{N'}\alpha_i - \frac{1}{2}\sum_{i,j=1}^{N'}\alpha_i\alpha_j y'_i y'_j K(\boldsymbol{\theta} * \mathbf{x}'_i, \boldsymbol{\theta} * \mathbf{x}'_j) \tag{3}$$

$$\text{subject to :} \quad 0 \leq \alpha_i \leq C, \qquad \sum_{i=1}^{N'}\alpha_i y'_i = 0$$

where the decision function $f_{\boldsymbol{\theta}}$ is given by $f_{\boldsymbol{\theta}}(\mathbf{x}) = \sum_{l=1}^{m}\alpha_l y_l K(\boldsymbol{\theta} * \mathbf{x}, \boldsymbol{\theta} * \mathbf{x}'_l) + b$, where $m$ is the number of support vectors. Eq. 3 is solved using a standard SVM algorithm [3, 19]. The optimization of the selection parameter $\boldsymbol{\theta}$ starts at the canonical solution where all dimensions are set to one. This is corresponding to the solution that is usually taken as a final model in other approaches. In our approach we apply a second optimization step to explicitly eliminate dimensions which are not necessary to classify data from the validation set. Fixing the values of the Lagrange multipliers $\boldsymbol{\alpha}$, the support vectors $\mathbf{x}'_l$ and the offset $b$ obtained by solving Eq. 3, leads to

$$\min_{\boldsymbol{\theta}} \lambda\|\boldsymbol{\theta}\|_1 + \frac{1}{2}\|\mathbf{w}\|_2^2 + C\sum_{i=1}^{N}\max(0, 1 - y_i f_{\boldsymbol{\theta}}(\mathbf{x}''_i)). \tag{4}$$

which is an instance of the regularized risk minimization problem $\min_{\boldsymbol{\theta}} \lambda\Omega(\boldsymbol{\theta}) + R(\boldsymbol{\theta})$, where $\Omega(\boldsymbol{\theta})$ is a regularization term and $R(\boldsymbol{\theta})$ is an upper bound on the empirical risk. To solve such non-differentiable risk minimization problems bundle methods have recently gained increasing interest in the machine learning community. For the case that the risk function $R$ is non-negative and convex

it is always lower bounded by its cutting plane at a certain point $\boldsymbol{\theta}^i$ :

$$R(\boldsymbol{\theta}) \geq <\mathbf{a}^i, \boldsymbol{\theta}> +b^i \text{ for all i} \qquad (5)$$

where $\mathbf{a}^i := \partial_{\boldsymbol{\theta}} R(\boldsymbol{\theta}^i)$ and $b^i := R(\boldsymbol{\theta}^i)- <\mathbf{a}^i, \boldsymbol{\theta}^i >$. Bundle methods build an iteratively increasing piecewise lower bound of the objective function by utilizing its cutting planes. Starting with an initial solution it solves the problem where $R$ is approximated by one initial cutting plane using standard solver. A second cutting plane is build at the solution of the approximated problem. The new approximated lower bound of $R$ is now the maximum over all cutting planes. The more cutting planes are added the more accurate gets the lower bound of the risk function.

For the general case of non-linear kernel functions the problem in Eq. 4 is a non-convex and therefore especially hard to optimize. In the special case of a linear kernel the problem is convex and the applied bundle method converges towards the global optimum. Some efforts have been made to adjust bundle methods to handle non-convex problems [16, 6]. We adapted the method of [6] to apply L1 regularization instead of L2 regularization and employ it to solve the optimization problem in Eq. 4. Although the convergence rate of $O(1/e)$ to a solution of accuracy $e$ [6] does no longer apply for our L1 regularized version, we observed that the algorithm converges withing the order of 10 iterations which is in the same range as for the algorithm in [6]. An overview of the suggested iterative dimensionality reduction algorithm is given in Algorithm 1.

## 3   Representing Curvature Self-Similarity

Although several methods have been suggested for the robust estimation of curvature, it has been mainly represented indirectly in a contour based manner [1, 32] and to locate interest points at boundary points with high curvature value. To design a more exact object representation that represents object curvedness in a natural way we revisit the idea of [21] and design a novel curvature self-similarity descriptor. The idea of self-similarity was first suggested by Shechtman et al. [25] who proposed a descriptor based on local self-similarity (LSS). Instead of measuring image features directly it measures the correlation of an image patch with a larger surrounding image region. The general idea of self-similarity was used in several methods and applications [5, 15, 29, 31]. In [15] self-similarity is used to improve the Local Binary Pattern (LBP) descriptor for face identification. Deselaers et al. [5] explored global self-similarity (GSS) and showed its advantages over local self-similarity (LSS) for object detection. Furthermore, Walk et al. [29] showed that using color histograms directly is decreasing performance while using color self-similarity (CSS) as a feature is more appropriate. Besides object classification and detection, self-similarity was also used for action recognition [15] and turned out to be very robust to viewpoint variations.
We propose a new holistic self-similarity representation based on curvature. To make use of the aforementioned advantages of global self-similarity we compute all pairwise curvature similarities across the whole image. This results in a very high dimensional object representation. As mentioned before such high dimensional representations have a natural need for dimensionality reduction which we fulfill by applying our embedded feature selection algorithm outlined in the previous section.

To describe complex objects it is not sufficient to build a self-similarity descriptor solely based on curvature information, since self-similarity of curvature leaves open many ambiguities. To resolve these ambiguities we add 360 degree orientation information to get a more accurate descriptor. We are using 360 degree orientation, since curved lines cannot be fully described by their 180 degree orientation. This is different to straight lines, where 180 degree orientation gives us the full information about the line. Consider a half circle, with an arbitrary tangent line on it. The tangent line has an orientation between 0 and 180 degrees. However, it does not provide information on which side of the tangent the half circle is actually located, in contrast to a 360 degree orientation. Therefore, using a 180 degree orientation yields to high similarities between a left curved line segment and a right curved line segment.

As a first step we extract the curvature information and the corresponding 360 degree orientation of all edge pixels in the image. To estimate the curvature we follow our approach presented in [21] and use the distance accumulation method of Han et al. [13], which accurately approximates the curvedness along given 2D line segments. Let $B$ be a set of $N$ consecutive boundary points, $B := \{p_0, p_1, p_2, ..., p_{N-1}\}$ representing one line segment. A fixed integer value $l$ defines a line $L_i$ between pairs of points $p_i$ to $p_{i+l}$, where $i + l$ is taken modulo $N$. The perpendicular distance $D_{ik}$

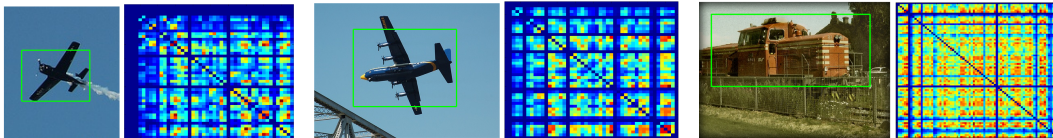

Figure 2: Our visualization shows the original images along with their curvature self-similarity matrices displaying the similarity between all pairs of curvature histogram cells. While curvature self-similarity descriptor is similar for the same object category it looks quite different to other object categories

is computed from $L_i$ to the point $p_k$, using the euclidean distance. The distance accumulation for point $p_k$ and a chord length $l$ is the sum $h_l(k) = \sum_{i=k-l}^{k} D_{ik}$ . The distance is positive if $p_k$ is on the left-hand side of the vector $(p_{i+l} - p_i)$, and negative otherwise (see Figure 1 and Figure 3). To get the 360 degree orientation information we compute the gradient of the probabilistic boundary edge image [20] and extend the resulting 180 degree gradient orientation to a 360 degree orientation using the sign of the curvature.

Contrary to the original curvature feature proposed in [21] where histograms of curvature are computed using differently sized image regions we build our basic curvature feature using equally sized cells to make it more suitable for computing self-similarities. We divide the image into non-overlapping $8 \times 8$ pixel cells and build histograms over the curvature values in each cell. Next we do the same for the 360 degree orientation and concatenate the two histograms. This results in histograms of 28 bins, 10 bins representing the curvature and 18 bins representing the 360 degree orientation. There are many ways to define similarities between histograms. We follow the scheme that was applied to compute self similarities between color histograms [29] and use histogram intersection as a comparison measure to compute the similarities between different curvature histograms in the same bounding box. Furthermore, we apply an L2-normalization to the final self-similarity vector. The computation of self-similarities between all curvature-orientation histograms results in an extremely high-dimensional representation. Let $D$ be the number of cells in an image, then computing all pairwise similarities results in a $D^2$ large curvature self-similarity matrix. Some examples are shown in Figure 2. Since, the similarity matrix is symmetric we use only the upper triangle which results in a $(D \cdot (D-1)/2)$-dimensional vector. This representation gives a very detailed description of the object.

The higher dimensional a descriptor gets, the more likely it contains noisy and correlated dimensions. Furthermore, it is also intuitive that not all similarities extracted from a bounding box are helpful to describe the object. To discard such superfluous dimensions we apply our embedded feature selection method to the proposed curvature self-similarity representation.

## 4 Experiments

We evaluate our curvature self-similarity descriptor in combination with the suggested embedded dimensionality reduction algorithm for the object detection task on the PASCAL dataset [7]. To show the individual strengths of these two contributions we need to perform a number of evaluations. Since this is not supported by the PASCAL VOC 2011 evaluation server we follow the best practice guidelines and use the VOC 2007 dataset. Our experiments show, that curvature self-similarity is providing complementary information to straight lines, while our feature selection algorithm is further improving performance by fulfilling its natural need for dimensionality reduction.

The common basic concept shared by many current detection systems are high-dimensional, holistic representations learned with a discriminative classifier, mostly an SVM [28]. In particular the combination of HOG [4] and SVM constitutes the basis of many powerful recognition systems and it has laid the foundation for numerous extensions like, part based models [8, 22, 24, 33], variations of the SVM classifier [8, 27] and approaches utilizing context information [14, 26]. These systems rely on high-dimensional holistic image statistics primarily utilizing straight line approximations. In this paper we explore a orthogonal direction to these extensions and focus on how one can improve on the basic system by extending the straight line representation of HOG to a more discriminative description using curvature self-similarity. At the same time our aim is to reduce the dimensionality

Table 1: Average precision of our iterative feature reduction algorithm for linear and non-linear kernel function using our final feature vector consisting of HOG+Curv+CurvSS. For linear kernel function we compare our feature selection (linSVM+FS) to L2 normalized linear SVM (linSVM) and to the doubly regularized SVM (DrSVM) [30]. For non-linear kernel function we compare the fast intersection kernel SVM (FIKSVM) [19] with our feature selection (FIKSVM+FS)

| | aero | bike | bird | boat | bottle | bus | car | cat | chair | cow |
|---|---|---|---|---|---|---|---|---|---|---|
| linSVM | 66.1 | 80.0 | 53.0 | 53.1 | 70.7 | 73.8 | 75.3 | 61.2 | 63.8 | 70.7 |
| DrSVM | 59.1 | 77.6 | 53.5 | 49.9 | 64.4 | 71.6 | 75.8 | 50.8 | 56.1 | 64.5 |
| linSVM + FS | 69.7 | 80.3 | 55.5 | 56.2 | 71.8 | 74.0 | 75.9 | 63.2 | 64.8 | 71.0 |
| FIKSVM | 80.1 | 74.8 | 57.1 | 59.3 | 63.3 | 73.9 | 77.3 | 77.3 | 69.1 | 66.4 |
| FIKSVM + FS | 80.4 | 74.9 | 57.5 | 62.1 | 66.7 | 73.9 | 78.0 | 80.1 | 70.6 | 69.9 |

| | table | dog | horse | mbike | pers | plant | sheep | sofa | train | tv | **mean** |
|---|---|---|---|---|---|---|---|---|---|---|---|
| linSVM | 71.4 | 57.2 | 76.5 | 83.0 | 72.9 | 47.7 | 55.1 | 61.1 | 70.4 | 73.1 | 66.8 |
| DrSVM | 59.9 | 53.9 | 70.9 | 76.5 | 72.3 | 47.7 | 66.3 | 69.0 | 67.7 | 79.7 | 64.3 |
| linSVM + FS | 72.0 | 57.8 | 77.2 | 83.3 | 73.0 | 49.7 | 56.7 | 62.4 | 70.7 | 73.8 | 68.0 |
| FIKSVM | 64.1 | 61.7 | 74.6 | 70.9 | 79.4 | 47.5 | 62.0 | 59.8 | 76.9 | 69.3 | 68.1 |
| FIKSVM + FS | 67.6 | 64.6 | 79.7 | 74.2 | 79.6 | 53.0 | 64.2 | 64.6 | 77.1 | 69.8 | 70.4 |

of such high-dimensional representations to decrease the complexity of the learning procedure and to improve generalization performance.

In the first part of our experiments we adjust the selection parameter $\lambda$ of our iterative dimensionality reduction technique via cross-validation. Furthermore, we compare the performance of our feature selection algorithm to L2 regularized SVM [3, 19] and DrSVM [30]. In the second part we evaluate the suggested curvature self-similarity feature after applying our feature selection method to it.

## 4.1   Evaluation of Feature Selection

All experiments in this section are performed using our final feature vector consisting of HOG, curvature (Curv) and curvature self-similarity (CurvSS). We apply our iterative dimensionality reduction algorithm in combination with linear L2 regularized SVM classifier (linSVM) [3] and non-linear fast intersection kernel SVM (FIKSVM) by Maji et al. [19]. The FIKSVM is widely used and evaluation is relatively fast compared to other non-linear kernels. Nevertheless, computational complexity is still an issue on the PASCAL dataset. This is why on this database linear kernels are typically used [8, 26].

Because of the high computational complexity of DrSVM and FIKSVM, we compare to these methods on a smaller train and test subset obtained from the PASCAL training and validation data in the following way. All training and validation data from the PASCAL VOC 2007 dataset are used to train an SVM using our final object representation on all positive samples and randomly chosen negative samples. The resulting model is used to collect hard negative samples. The set of collected samples is split up into three sets: training, validation and test. Out of the collected set of samples every tenth sample is assigned to the hold out test set which is used to compare the performance of our feature selection method. The remaining samples are randomly split into training and validation set of equal size which are used to perform the feature selection. The reduction algorithm is applied on 5 different training/validation splits which results in five different sets of selected features. For each set we train an L2 norm SVM on all samples from the training and validation set using only the remaining dimensions of the feature vector. Then we choose the feature set with the best performance on the hold out test set. To find the best performing selection parameter $\lambda$, we repeat this procedure for different values of $\lambda$.

The performance of our dimensionality reduction algorithm is compared to the performance of linSVM and DrSVM [30] for the case of a linear kernel. Since DrSVM is solving a similar optimization problem as our suggested feature selection algorithm for a linear kernel this comparison is of particular interest. We are not comparing performance to DrSVM in the non-linear case since

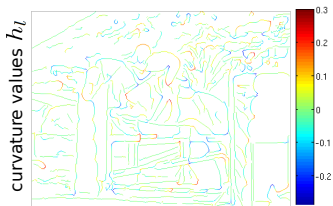

Figure 3: Based on meaningful edge images one can extract accurate curvature information which is used to build our curvature self-similarity object representation

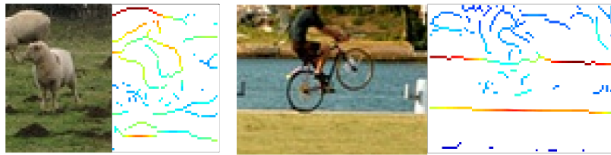

Figure 4: A significant number of images from PASCAL VOC feature contour artifacts i.e, due to their size, low resolution, or compression artifacts. The edge maps are obtained from the state-of-the-art probabilistic boundary detector [20]. It is evident that objects like the sheep are not defined by their boundary shape and are thus beyond the scope of approaches base on contour shape

it is performing feature selection in the higher dimensional kernel space rather than in the original feature space. Instead we compare our feature selection method to that of FIKSVM for the non-linear case. Our feature selection method reduces the dimensionality of the feature by up to $55\%$ for the linear case and by up to $40\%$ in the non-linear case, while the performance in average precision is constant or increases beyond the performance of linSVM and FIKSVM. On average our feature selection increases performance about $1.2\%$ for linSVM and $2.3\%$ for FIKSVM on the hold-out testset. The DrSVM is actually decreasing the performance of linSVM by $2.5\%$ while discarding a similar amount of features. All in all our approach improves the DrSVM by $3.7\%$ (see Table 1). Our results confirm that our feature selection method reduces the amount of noisy dimensions of high-dimensional representations and therefore increases the average precision compared to an linear and non-linear SVM classifier without applying any feature selection. For the linear kernel we showed furthermore that the proposed feature selection algorithm achieves gain over the DrSVM.

## 4.2 Object Detection using Curvature Self-Similarity

In this section we provide a structured evaluation of the parts of our final object detection system. We use the HOG of Felzenszwalb et al. [8, 9] as baseline system, since it is the basis for many powerful object detection systems. All detection results are measured in terms of average precision performing object detection on the PASCAL VOC 2007 dataset.

To the best of our knowledge neither curvature nor self-similarity was used to perform object detection on a dataset of similar complexity as the PASCAL dataset so far. Deselaers et al. [5] evaluated their global self-similarity descriptor (GSS) on the simpler classification challenge on the PASCAL VOC 2007 dataset, while the object detection evaluation was performed on the ETHZ shape dataset. However, we showed in [21], that including curvature already solves the detection task almost perfectly on the ETHZ dataset. Furthermore, [21] outperforms the GSS descriptor on three categories and reached comparable performance on the other two. Thus we evaluate on the more challenging PASCAL dataset. Since the proposed approach models the shape of curved object contours and reduces the dimensionality of the representation, we expect it to be of particular value for objects that are characterized by their shape and where their contours can be extracted using state-of-the-art methods. However, a significant number of images form PASCAL VOC are corrupted due to noise or compression artifacts (see Fig. 4). Therefore state-of-the-art edge extraction fails to provide any basis for contour based approaches on these images and one can therefore only expect a significant gain on categories where proper edge information can be computed for a majority of the images.

Our training procedure makes use of all objects that are not marked as difficult from the training and validation set. We evaluate the performance of our system on the full testset consisting of 4952 images containing objects from 20 categories using a linear SVM classifier [3]. Due to the large amount of data in the PASCAL database the usage of intersection kernel for object detection becomes comparable intractable. Results of our final system consisting of HOG, curvature (Curv), curvature self-similarity (CurvSS) and our embedded feature selection method (FS) are reported in terms of average precision in Table 2. We compare our results to that of HOG [9] without applying the part based model. Additionally we show results of our own HOG baseline system which is using standard linear SVM [3] instead of the latent SVM used in [9]. Furthermore we show results with

Table 2: Detection performance in terms of average precision of the HOG baseline system, HOG and curvature (Curv) before and after discarding noisy dimensions using our feature selection method (FS) and our final detection system consisting of HOG, curvature (Curv), the suggested curvature self-similarity (CurvSS) with and without feature selection (FS) on the PASCAL VOC 2007 dataset. Note, that we use all data points to compute the average precision as it is specified by the default experimental protocol since VOC 2010 development kit. This yields lower but more accurate average precision measurements

|  | aero | bike | bird | boat | bottle | bus | car | cat | chair | cow |
|---|---|---|---|---|---|---|---|---|---|---|
| HOG of [9] | 19.0 | *44.5* | 2.9 | 4.2 | 13.5 | 37.7 | 39.0 | 8.3 | 11.4 | 15.8 |
| HOG | 20.8 | 43.0 | 2.1 | 5.0 | *13.7* | 37.8 | 38.7 | 6.7 | 12.1 | 16.3 |
| HOG+Curv | 23.0 | 42.6 | *3.7* | 6.7 | 12.4 | 38.6 | 39.9 | 7.5 | 10.0 | 16.9 |
| HOG+Curv+FS | 25.4 | 42.9 | *3.7* | 6.8 | 13.5 | 38.8 | 40.0 | 8.1 | 12.0 | 17.1 |
| HOG+Curv+CurvSS | 28.6 | 39.1 | 2.3 | 6.8 | 12.9 | 40.3 | 38.8 | 9.3 | 11.1 | 13.9 |
| HOG+Curv+CurvSS+FS | *28.9* | 43.1 | 3.5 | *7.0* | 13.6 | *40.6* | *40.4* | *9.6* | *12.5* | *17.3* |

|  | table | dog | horse | mbike | pers | plant | sheep | sofa | train | tv | **mean** |
|---|---|---|---|---|---|---|---|---|---|---|---|
| HOG of [9] | 10.5 | 2.0 | 43.5 | 29.7 | 24.0 | 3.0 | *11.6* | 17.7 | 28.3 | 32.4 | 20.0 |
| HOG | 9.8 | 2.2 | 42.4 | 29.5 | 24.3 | 3.8 | 11.5 | 17.6 | 29.0 | 33.4 | 20.0 |
| HOG+Curv | 13.0 | 3.7 | 46.0 | 30.5 | 25.5 | 4.0 | 8.7 | 18.7 | 32.3 | 33.6 | 20.9 |
| HOG+Curv+FS | 15.6 | 3.7 | 46.4 | *30.8* | 25.7 | 4.0 | 11.3 | 19.1 | 32.3 | 33.6 | 21.5 |
| HOG+Curv+CurvSS | 16.3 | 6.2 | 48.0 | 27.5 | 27.2 | 4.2 | 9.3 | 20.5 | 35.9 | *34.8* | 21.7 |
| HOG+Curv+CurvSS+FS | *16.7* | *6.4* | *48.5* | 30.6 | *27.3* | *4.8* | *11.6* | *20.7* | *36.0* | *34.8* | *22.7* |

and without feature selection to show the individual gain of the curvature self-similarity descriptor and our embedded feature selection algorithm.

The results show that the suggested self-similarity representation in combination with feature selection improves performance on most of the categories. All in all this results in an increase of $2.7\%$ in average precision compared to the HOG descriptor. One can observe that curvature information in combination with our feature selection algorithm is already improving performance over the HOG baseline and that adding curvature self-similarity additionally increases performance by $1.2\%$. The gain obtained by applying our feature selection (FS) depends obviously on the dimensionality of the feature vector; the higher the dimensionality the more can be gained by removing noisy dimensions. For HOG+Curv applying our feature selection is improving performance by $0.6\%$ while the gain for the higher dimensional HOG+Curv+CurvSS is $1\%$. The results underline that curvature information provides complementary information to straight lines and that feature selection is needed when dealing with high dimensional features like self-similarity.

## 5 Conclusion

We have observed that high-dimensional representations cannot be sufficiently handled by linear and non-linear SVM classifiers. An embedded feature selection method for SVMs has therefore been proposed in this paper, which has been demonstrated to successfully deal with high-dimensional descriptions and it increases the performance of linear and intersection kernel SVM. Moreover, the proposed curvature self-similarity representation has been shown to add complementary information to widely used orientation histograms.[1]

## References

[1] S. Belongie, J. Malik, and J. Puzicha. Matching shapes. *ICCV*, 2001.

[1]This work was supported by the Excellence Initiative of the German Federal Government and the Frontier fund, DFG project number ZUK 49/1.

[2] P. S. Bradley and O. L. Magasarian. Feature selection via concave minimization and support vector machines. *ICML*, 1998.

[3] C.-C Chang and C.-J. Lin. LIBSVM: A library for support vector machines. *ACM Transactions on Intelligent Systems and Technology*, 2:27:1–27:27, 2011.

[4] N. Dalal and B. Triggs. Histograms of oriented gradients for human detection. *CVPR*, 2005.

[5] T. Deselaers and V. Ferrari. Global and efficient self-similarity for object classification and detection. *CVPR*, 2010.

[6] T.-M.-T. Do and T. Artiéres. Large margin training for hidden markov models with partially observed states. *ICML*, 2009.

[7] M. Everingham, L. Van Gool, C. K. I. Williams, J. Winn, and A. Zisserman. The PASCAL Visual Object Classes Challenge 2007 (VOC2007) Results. http://www.pascal-network.org/challenges/VOC/voc2007/workshop/index.html.

[8] P. Felzenszwalb, R. Girshick, D. McAllester, and D. Ramanan. Object detection with discriminatively trained part based models. *PAMI*, 2010.

[9] P. F. Felzenszwalb, R. B. Girshick, and D. McAllester. Discriminatively trained deformable part models, release 4. http://www.cs.brown.edu/ pff/latent-release4/.

[10] W. T. Freeman and M. Roth. Orientation histograms for hand gesture recognition. *Intl. Workshop on Automatic Face and Gesture- Recognition*, 1995.

[11] Y. Grandvalet and S. Canu. Adaptive scaling for feature selection in SVMs. *NIPS*, 2003.

[12] I. Guyon and A. Elisseeff. An introduction to variable and feature selection. *JMLR*, 3:11571182, 2003.

[13] J. H. Han and T. Poston. Chord-to-point distance acccumulation and planar curvature: a new approach to discrete curvature. *Pattern Recognition Letters*, 22(10):1133 – 1144, 2001.

[14] G. Heitz and D. Koller. Learning spatial context: Using stuff to find things. *ECCV*, 2008.

[15] I. N. Junejo, E. Dexter, I. Laptec, and P. Peréz. Cross-view action recognition from temporal self-similarities. *ECCV*, 2008.

[16] N. Karmitsa, M. Tanaka Filho, and J. Herskovits. Globally convergent cutting plane method for nonconvex nonsmooth minimization. *Journal of Optimization Theory and Applications*, 148(3):528 – 549, 2011.

[17] T. N. Lal, O. Chapelle, J. Weston, and A. Elisseeff. *Studies in Fuzziness and Soft Computing*. I. Guyon and S. Gunn and N. Nikravesh and L. A. Zadeh, 2006.

[18] D.G. Lowe. Object recognition from local scale-invariant features. *ICCV*, 1999.

[19] S. Maji, A. C. Berg, and J. Malik. Classification using intersection kernel support vector machines is efficient. *CVPR*, 2008.

[20] D. Martin, C. Fowlkes, and J. Malik. Learning to detect natural image boundaries using local brightness, color, and texture cues. *PAMI*, 26(5):530 – 549, 2004.

[21] A. Monroy, A. Eigenstetter, and B. Ommer. Beyond straight lines - object detection using curvature. *ICIP*, 2011.

[22] A. Monroy and B. Ommer. Beyond bounding-boxes: Learning object shape by model-driven grouping. *ECCV*, 2012.

[23] C. P. Papageorgiou, M. Oren, and T. Poggio. A general framwork for object detection. *ICCV*, 1998.

[24] P. Schnitzspan, M. Fritz, S. Roth, and B. Schiele. Discriminative structure learning of hierarchical representations for object detection. *CVPR*, 2009.

[25] E. Shechtman and M. Irani. Matching local self-similarities across images and videos. *CVPR*, 2007.

[26] Z. Song, Q. Chen, Z. Huang, Y. Hua, and S. Yan. Contextualizing object detection and classification. *CVPR*, 2011.

[27] I. Tsochantaridis, T. Hofmann, T. Joachims, and Y. Altun. Support vector learning for interdependent and structured output spaces. *ICML*, 2004.

[28] V. N. Vapnik. *The Nature of Statistical Learning Theory*. Springer Verlag, 1995.

[29] S. Walk, N. Majer, K. Schindler, and B. Schiele. New features and insights for pedestiran detection. *CVPR*, 2010.

[30] L. Wang, J. Zhu, and H. Zou. The doubly regularized support vector machine. *Statistica Sinica*, 16, 2006.

[31] L. Wolf, T. Hassner, and Y. Taigman. Descriptor based methods in the wild. *ECCV*, 2008.

[32] P. Yarlagadda and B. Ommer. From meaningful contours to discriminative object shape. *ECCV*, 2012.

[33] L. Zhu, Y. Chen, A. Yuille, and W. Freeman. Latent hierarchical structural learning for object detection. *CVPR*, pages 1062 –1069, 2010.

